# DISCOVERING STRUCTURE FROM MOTION IN MONKEY, MAN AND MACHINE

Ralph M. Siegel*

The Salk Institute of Biology, La Jolla, Ca. 92037

## ABSTRACT

The ability to obtain three-dimensional structure from visual motion is important for survival of human and non-human primates. Using a parallel processing model, the current work explores how the biological visual system might solve this problem and how the neurophysiologist might go about understanding the solution.

## INTRODUCTION

Psychophysical experiments have shown that monkey and man are equally adept at obtaining three dimensional structure from motion[1]. In the present work, much effort has been expended mimicking the visual system. This was done for one main reason: the model was designed to help direct physiological experiments in the primate. It was hoped that if an approach for understanding the model could be developed, the approach could then be directed at the primate's visual system.

Early in this century, von Helmholtz[2] described the problem of extracting three-dimensional structure from motion:

Suppose, for instance, that a person is standing still in a thick woods, where it is impossible for him to distinguish, except vaguely and roughly, in the mass of foliage and branches all around him what belongs to one tree and what to another, or how far apart the separate trees are, etc. But the moment he begins to move forward, everything disentangles itself, and immediately he gets an apperception of the material content of the woods and their relation to each other in space, just as if he were looking at a good stereoscopic view of it.

If the object moves, rather than the observer, the perception of three-dimensional structure from motion is still obtained. Object-centered structure from motion is examined in this report. Lesion studies in monkey have demonstrated that two extra-striate visual cortices called the middle temporal area (abbreviated MT

or V5) and the medial superior temporal area (MST)[3,4] are involved in obtaining structure from motion. The present model is meant to mimic the V5-MST part of the cortical circuitry involved in obtaining structure from motion. The model attempts to determine if the visual image corresponds to a three-dimensional object.

## THE STRUCTURE FROM MOTION STIMULUS

The problem that the model solved was the same as that posed in the studies of monkey and man[1]. Structured and unstructured motion displays of a hollow, orthographically projected cylinder were computed (Figure 1). The cylinder rotates about its vertical axis. The unstructured stimulus was generated by shuffling the velocity vectors randomly on the display screen. The overall velocity and spatial distribution for the two displays are identical; only the spatial relationships have been changed in the unstructured stimulus. Human subjects report that the points are moving on the surface of a hollow cylinder when viewing the structured stimulus. With the unstructured stimulus, most subjects report that they have no sense of three-dimensional structure.

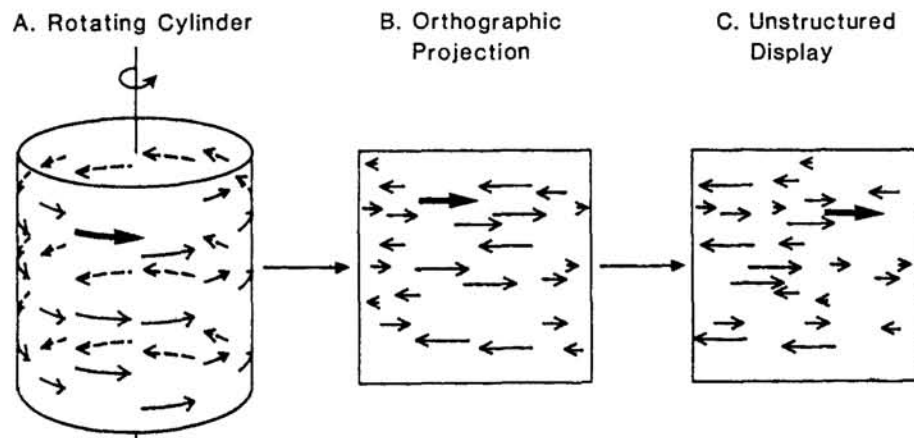

A. Rotating Cylinder    B. Orthographic Projection    C. Unstructured Display

Figure 1. The structured and unstructured motion stimulus. A) "N" points are randomly placed on the surface of a cylinder. B) The points are orthographically projected. The motion gives a strong percept of a hollow cylinder. C) The unstructured stimulus was generated by shuffling the velocity vectors randomly on the screen.

## FUNCTIONAL ARCHITECTURE OF THE MODEL

As with the primate subjects, the model was required to only indicate whether or not the display was structured. Subjects were not required to describe the shape, velocity or size of the cylinder. Thus the output cell* of the model signaled "1" if

structured and "0" if not structured. This output layer corresponds to the cortical area MST of macaque monkey which appear to be sensitive to the global organization of the motion image[5]. It is not known if MST neurons will distinguish between structured and unstructured images.

The input to the model was based on physiological studies in the macaque monkey. Neurons in area V5 have a retinotopic representation of visual space[6,7].

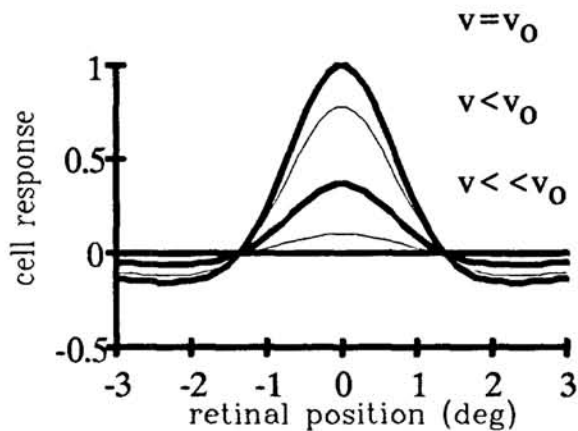

Figure 2. The receptive field of an input layer cell. The optimal velocity is "$v_o$".

For each retinotopic location there is an encoding of a wide range of velocities[8]. Thus in the model's input representation, there were cells that represent different combinations of velocity and retinotopic spatial position. Furthermore motion velocity neurons in V5 have a center-surround opponent organization[9]. The width of the receptive fields was taken from the data of Albright et al.[8]. A typical receptive field of the model is shown in Figure 2.

It was possible to determine what the activity of the input cells would be for the rotating cylinder given this representation. The activation pattern of the set of input cells was computed by convolving the velocity points with the difference of gaussians. The activity of the 100 input cells for an image of 20 points, with an angular velocity of $8^o$/sec is presented in Figure 3.

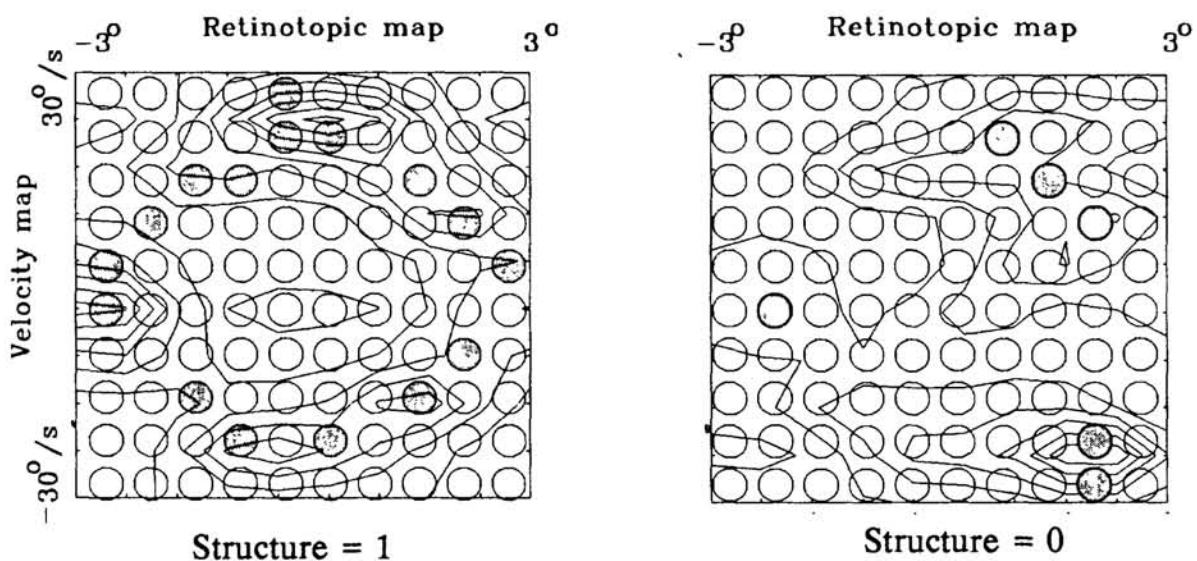

Figure 3. The input cell's activation pattern for a structured and unstructured stimulus. The circles correspond to the cells of the input layer. The contours were com-

puted using a linear interpolation between the individual cells. The horizontal axis corresponds to the position along the horizontal meridian. The vertical axis corresponds to the speed along the horizontal meridian. Thus activation of a cell in the upper right hand corner of the graph correspond to a velocity of $30^{\circ}$/sec towards the right at a location of $3^{\circ}$ to the right along the horizontal meridian.

Inspection of this input pattern suggested that the problem of detecting three-dimensional structure from motion may be reduced to a pattern recognition task. The problem was then: "Given a sparsely sampled input motion flow field, determine whether it corresponds best to a structured or unstructured object."

It was next necessary to determine the connections between the two input and output layers such that the model will be able to correctly signal structure or no structure (1 or 0) over a wide range of cylinder radii and rotational velocities. A parallel distributed network of the type used by Rosenberg and Sejnowski[10] provided the functional architecture (Figure 4).

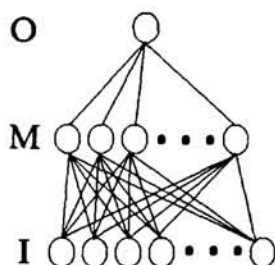

Figure 4. The parallel architecture used to extract structure from motion. The input layer (I), corresponding to area V5, mapped the position and speed along the horizontal axis. The output layer (O) corresponded to area MST that, it is proposed, signals structure or not. The middle layer (M) may exist in either V5 or MST.

The input layer of cells was fully connected to the middle layer of cells. The middle layer of cells represented an intermediate stage of processing and may be in either V5 or MST. All of the cells of the middle layer were then fully connected to the output cell. The inputs from cells of the lower layer to the next higher level were summed linearly and then "thresholded" using the Hill equation $X^3/(X^3+0.5^3)$. The weights between the layers were initially chosen between $\pm 1$. The values of the weights were then adjusted using back-propagation methods (steepest descent) so that the network would "learn" to correctly predict the structure of the input image. The model learned to correctly perform the task after about 10,000 iterations (Figure 5).

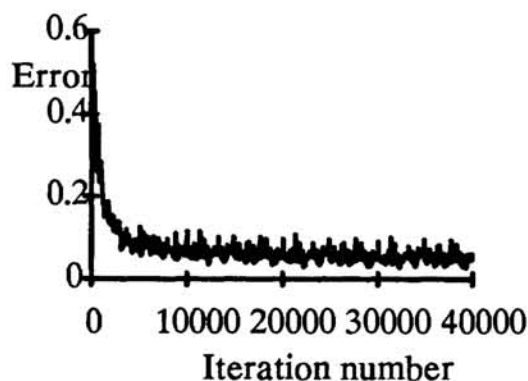

Iteration number

Figure 5. The "education" of the network to perform the structure from motion problem. The iteration number is plotted against the mean square error. The error is defined as the difference between the model's prediction and the known structure. The model was trained on a set of structured and unstructured cylinders with a wide range of radii, number of points, and rotational velocities.

## PSYCHOPHYSICAL PERFORMANCE OF THE MODEL

The model's performance was comparable to that of monkey and man with respect to fraction of structure and number of points in the display (Figure 6). The model was indeed performing a global analysis as shown by allowing the model to view only a portion of the image. Like man and monkey, the model's performance suffers. Thus it appears that the model's performance was quite similar to known monkey and human psychophysics.

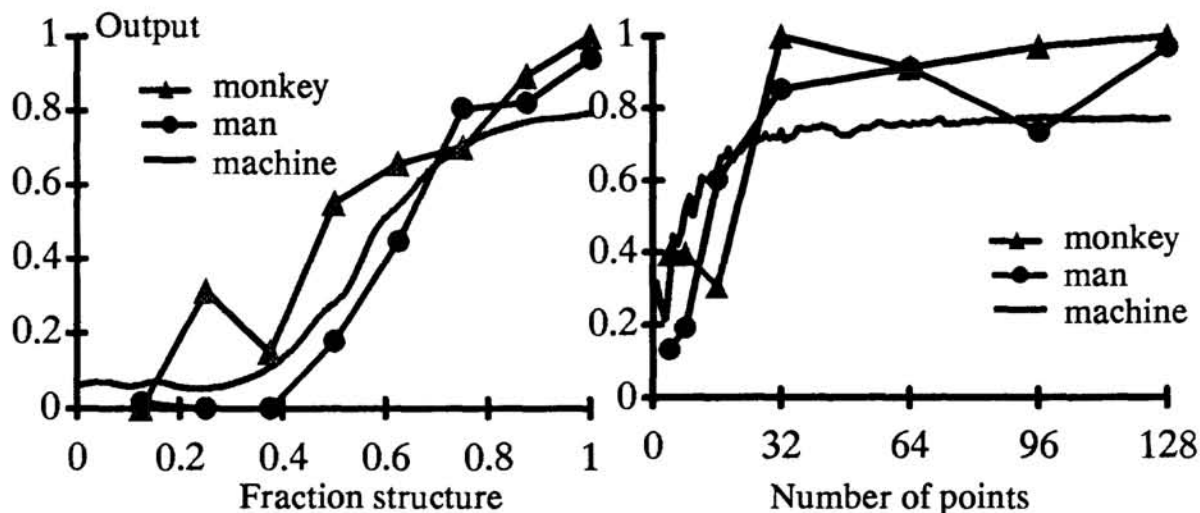

Figure 6. Psychophysical performance of the model. A. The effect of varying the fraction of structure. As the fraction of structure increase, the model's performance improves. Thirty repetitions were averaged for each value of structure for the model. The fraction of structure is defined as $(1-R_s/R_c)$, where $R_s$ is the radius of shuffling of the motion vectors and $R_c$ is the radius of the cylinder. The human and monkey data are taken from psychophysical studies[1].

## HOW IS IT DONE?

The model has similar performance to monkey and man. It was next possible to examine this artificial network in order to obtain hints for studying the biological system. Following the approach of an electrophysiologist, receptive field maps for all the cells of the middle and output layers were made by activating individual input cells. The receptive field of some middle layer cells are shown in Figure 7. The layout of these maps are quite similar to that of Figure 4. However, now the activity of one cell in the middle layer is plotted as a function of the location and speed of a motion stimulus in the input layer. One could imagine that an electrode was placed in one of the cells of the middle layer while the experimentalist moved a bar about the horizontal meridian with different locations and speeds. The activity of the cell is then plotted as a function of position and space.

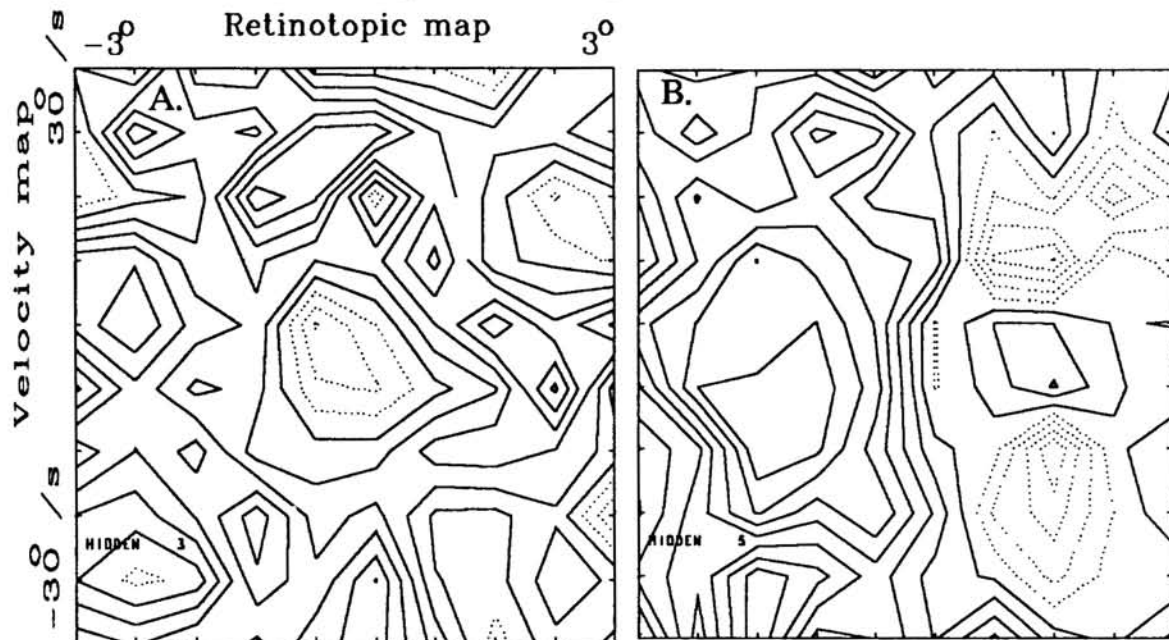

Figure 7. The activity of two different cells in the middle layer. Activity is plotted as a contour map as a function of horizontal position and speed. Dotted lines indicate inhibition.

These middle layer receptive field maps were interesting because they appear to be quite simple and symmetrical. In some, the inhibitory central regions of the receptive field were surrounded by excitatory regions (Figure 7A). Complementary cells were also found. In others, there are inhibitory bands adjacent to excitatory bands (Figure 7B). The above results suggest that neurons involved in extracting structure from motion may have relatively simple receptive fields in the spatial velocity domain. These receptive fields might be thought of as breaking the image down into component parts (i.e. a basis set). Correct recombination of these second order cells could then be used to detect the presence of a three-dimensional structure.

The output cell also had a simple receptive field again with interesting symmetries (Figure 8). However, the receptive field analysis is insufficient to indicate the role of the cell. Therefore in order to properly understand the "mean-

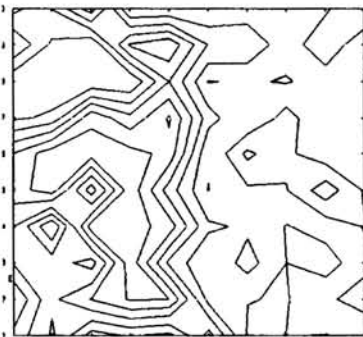

ing" of the cell's receptive field, it is necessary to use stimuli that are "real world relevant"- in this case the structure from motion stimuli. The output cell would give its maximal response only when a cylinder stimulus is presented.

Figure 8. The receptive field map of the output layer cell. Nothing about this receptive field structure indicates the cell is involved in obtaining structure from motion.

This work predicts that neurons in cortex involved in extracting structure from motion will have relatively simple receptive fields. In order to test this hypothesis, it will be necessary to make careful maps of these cells using small patches of motion (Figure 9). Known qualitative results in areas V5 and MST are consistent with, but do not prove, this hypothesis. As well, it will be necessary to use "relevant" stimuli (e.g. three-dimensional objects). If such simple receptive fields are indeed used in structure from motion, then support will be found for the idea that a simple cortical circuit (e.g. center-surround) can be used for many different visual analyses.

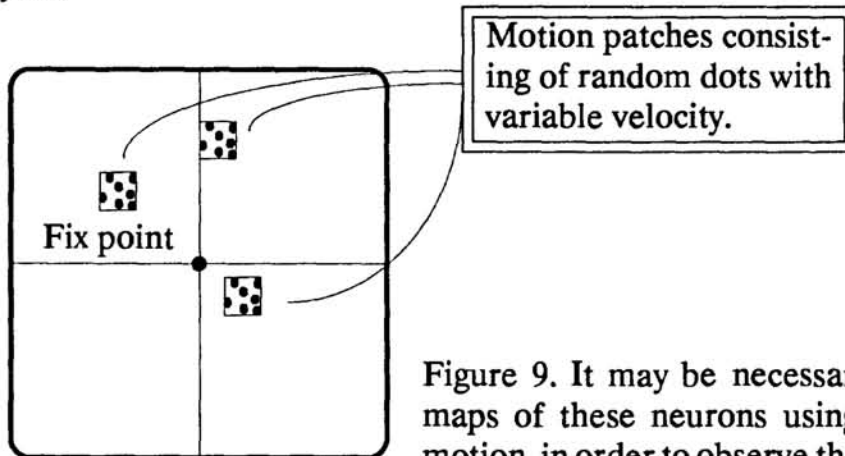

Motion patches consisting of random dots with variable velocity.

Figure 9. It may be necessary to make careful maps of these neurons using small patches of motion, in order to observe the postulated simple receptive field properties of cortical neurons involved in extracting structure from motion. Such structures may not be apparent using hand moved bar stimuli.

## DISCUSSION

In conclusion, it is possible to extract the three-dimensional structure of a rotating cylinder using a parallel network based on a similar functional architecture as found in primate cortex. The present model has similar psychophysics to monkey and man. The receptive field structures that underlie the present model are simple when viewed using a spatial-velocity representation. It is suggested that in order to understand how the visual system extracts structure from motion, quantitative spatial-velocity maps of cortical neurons involved need to be made. One also needs to use stimuli derived from the "real world" in order to understand how they may be used in visual field analysis. There are similarities between the shapes of the receptive fields involved in analyzing structure from motion and receptive fields in striate cortex[11]. It may be that similar cortical mechanisms and connections are used to perform different functions in different cortical areas. Lastly, this model demonstrates that the use of parallel architectures that are closely modeled on the cortical representation is a computationally efficient means to solve problems in vision. Thus as a final caveat, I would like to advise the creators of networks that solve ethologically realistic problems to use solutions that evolution has provided.

## Footnotes

*Current address: Laboratory of Neurobiology, The Rockefeller University, 1230 York Avenue, New York, NY 10021

*By cell, I mean a processing unit of the model which may correspond to a single neuron or group of neurons. The term neuron refers only to the actual wetware in the brain.

## REFERENCES

1. R.M. Siegel and R.A. Andersen, Nature (Lond.) (1988).

2. H. von Helmholtz, Treatise on Physiological Optics (Dover Publications, N.Y., 1910), p. 297.

3. R.M. Siegel and R.A. Andersen, Soc. Neurosci. Abstr., 12, p. 1183 (1986).

4. R.M. Siegel and R.A. Andersen, Localization of function in extra-striate cortex: the effect of ibotenic acid lesions on motion sensitivity in Rhesus monkey, (in preparation).

5. K. Tanaka, K. Hikosaka, H. Saito, M. Yukie, Y. Fukada, and E. Iwai, J., Neurosci., 6, pp. 134-144 (1986).

6. S.M. Zeki, Brain Res., 35, pp. 528-532 (1971).

7. J.H.R. Maunsell and D.C. Van Essen, J. Neurophysiol., 49, pp. 1127-1147 (1983).

8. T.D. Albright, R. Desimone, and C.G. Gross, J. Neurophysiol., 51, pp. 16-31 (1984).

9. J. Allman, F. Miezen, and E. McGuinness, Ann. Rev. Neurosci., 8, pp. 407-430 (1985).

10. C.R. Rosenberg and T.J. Sejnowski, in: Reports of the Cognitive Neuropsychology Laboratory, John-Hopkins University (1986).

11. D.H. Hubel and T.N. Wiesel, Proc. R. Soc. Lond. B., 198, pp.1-59 (1977).

This work was supported by the Salk Institute for Biological Studies, The San Diego Supercomputer Center, and PHS NS07457-02.
